# LSTD with Random Projections

**Mohammad Ghavamzadeh, Alessandro Lazaric, Odalric-Ambrym Maillard, Rémi Munos**
INRIA Lille - Nord Europe, Team SequeL, France

## Abstract

We consider the problem of reinforcement learning in high-dimensional spaces when the number of features is bigger than the number of samples. In particular, we study the least-squares temporal difference (LSTD) learning algorithm when a space of low dimension is generated with a random projection from a high-dimensional space. We provide a thorough theoretical analysis of the LSTD with random projections and derive performance bounds for the resulting algorithm. We also show how the error of LSTD with random projections is propagated through the iterations of a policy iteration algorithm and provide a performance bound for the resulting least-squares policy iteration (LSPI) algorithm.

## 1 Introduction

Least-squares temporal difference (LSTD) learning [3, 2] is a widely used reinforcement learning (RL) algorithm for learning the value function $V^\pi$ of a given policy $\pi$. LSTD has been successfully applied to a number of problems especially after the development of the least-squares policy iteration (LSPI) algorithm [9], which extends LSTD to control problems by using it in the policy evaluation step of policy iteration. More precisely, LSTD computes the fixed point of the operator $\Pi \mathcal{T}^\pi$, where $\mathcal{T}^\pi$ is the Bellman operator of policy $\pi$ and $\Pi$ is the projection operator onto a linear function space. The choice of the linear function space has a major impact on the accuracy of the value function estimated by LSTD, and thus, on the quality of the policy learned by LSPI. The problem of finding the right space, or in other words the problems of feature selection and discovery, is an important challenge in many areas of machine learning including RL, or more specifically, linear value function approximation in RL.

To address this issue in RL, many researchers have focused on feature extraction and learning. Mahadevan [13] proposed a constructive method for generating features based on the eigenfunctions of the Laplace-Beltrami operator of the graph built from observed system trajectories. Menache et al. [16] presented a method that starts with a set of features and then tunes both features and the weights using either gradient descent or the cross-entropy method. Keller et al. [7] proposed an algorithm in which the state space is repeatedly projected onto a lower dimensional space based on the Bellman error and then states are aggregated in this space to define new features. Finally, Parr et al. [17] presented a method that iteratively adds features to a linear approximation architecture such that each new feature is derived from the Bellman error of the existing set of features.

A more recent approach to feature selection and discovery in value function approximation in RL is to solve *RL in high-dimensional feature spaces*. The basic idea here is to use a large number of features and then exploit the regularities in the problem to solve it efficiently in this high-dimensional space. Theoretically speaking, increasing the size of the function space can reduce the approximation error (the distance between the target function and the space) at the cost of a growth in the estimation error. In practice, in the typical high-dimensional learning scenario when the number of features is larger than the number of samples, this often leads to the overfitting problem and poor prediction performance. To overcome this problem, several approaches have been proposed including regularization. Both $\ell_1$ and $\ell_2$ regularizations have been studied in value function approximation in RL. Farahmand et al. presented several $\ell_2$-regularized RL algorithms by adding $\ell_2$-regularization to LSTD and modified Bellman residual minimization [4] as well as fitted value iteration [5], and proved finite-sample performance bounds for their algorithms. There have also been algorithmic work on adding $\ell_1$-penalties to the TD [12], LSTD [8], and linear programming [18] algorithms.

In this paper, we follow a different approach based on random projections [21]. In particular, we study the performance of *LSTD with random projections* (LSTD-RP). Given a high-dimensional linear space $\mathcal{F}$, LSTD-RP learns the value function of a given policy from a small (relative to the dimension of $\mathcal{F}$) number of samples in a space $\mathcal{G}$ of lower dimension obtained by linear random projection of the features of $\mathcal{F}$. We prove that solving the problem in the low dimensional random space instead of the original high-dimensional space reduces the estimation error at the price of a "controlled" increase in the approximation error of the original space $\mathcal{F}$. We present the LSTD-RP algorithm and discuss its computational complexity in Section 3. In Section 4, we provide the finite-sample analysis of the algorithm. Finally in Section 5, we show how the error of LSTD-RP is propagated through the iterations of LSPI.

## 2   Preliminaries

For a measurable space with domain $\mathcal{X}$, we let $\mathcal{S}(\mathcal{X})$ and $\mathcal{B}(\mathcal{X}; L)$ denote the set of probability measures over $\mathcal{X}$ and the space of measurable functions with domain $\mathcal{X}$ and bounded in absolute value by $0 < L < \infty$, respectively. For a measure $\mu \in \mathcal{S}(\mathcal{X})$ and a measurable function $f : \mathcal{X} \to \mathbb{R}$, we define the $\ell_2(\mu)$-norm of $f$ as $||f||_\mu^2 = \int f(x)^2 \mu(dx)$, the supremum norm of $f$ as $||f||_\infty = \sup_{x \in \mathcal{X}} |f(x)|$, and for a set of $n$ states $X_1, \ldots, X_n \in \mathcal{X}$ the empirical norm of $f$ as $||f||_n^2 = \frac{1}{n} \sum_{t=1}^{n} f(X_t)^2$. Moreover, for a vector $u \in \mathbb{R}^n$ we write its $\ell_2$-norm as $||u||_2^2 = \sum_{i=1}^{n} u_i^2$.

We consider the standard RL framework [20] in which a learning agent interacts with a stochastic environment and this interaction is modeled as a discrete-time discounted Markov decision process (MDP). A discount MDP is a tuple $\mathcal{M} = \langle \mathcal{X}, \mathcal{A}, r, P, \gamma \rangle$ where the state space $\mathcal{X}$ is a bounded closed subset of a Euclidean space, $\mathcal{A}$ is a finite ($|\mathcal{A}| < \infty$) action space, the reward function $r : \mathcal{X} \times \mathcal{A} \to \mathbb{R}$ is uniformly bounded by $R_{\max}$, the transition kernel $P$ is such that for all $x \in \mathcal{X}$ and $a \in \mathcal{A}$, $P(\cdot|x, a)$ is a distribution over $\mathcal{X}$, and $\gamma \in (0, 1)$ is a discount factor. A deterministic policy $\pi : \mathcal{X} \to \mathcal{A}$ is a mapping from states to actions. Under a policy $\pi$, the MDP $\mathcal{M}$ is reduced to a Markov chain $\mathcal{M}^\pi = \langle \mathcal{X}, R^\pi, P^\pi, \gamma \rangle$ with reward $R^\pi(x) = r(x, \pi(x))$, transition kernel $P^\pi(\cdot|x) = P(\cdot|x, \pi(x))$, and stationary distribution $\rho^\pi$ (if it admits one). The value function of a policy $\pi$, $V^\pi$, is the unique fixed-point of the Bellman operator $\mathcal{T}^\pi : \mathcal{B}(\mathcal{X}; V_{\max} = \frac{R_{\max}}{1-\gamma}) \to \mathcal{B}(\mathcal{X}; V_{\max})$ defined by $(\mathcal{T}^\pi V)(x) = R^\pi(x) + \gamma \int_{\mathcal{X}} P^\pi(dy|x)V(y)$. We also define the optimal value function $V^*$ as the unique fixed-point of the optimal Bellman operator $\mathcal{T}^* : \mathcal{B}(\mathcal{X}; V_{\max}) \to \mathcal{B}(\mathcal{X}; V_{\max})$ defined by $(\mathcal{T}^* V)(x) = \max_{a \in \mathcal{A}} \left[ r(x, a) + \gamma \int_{\mathcal{X}} P(dy|x, a)V(y) \right]$. Finally, we denote by $T$ the truncation operator at threshold $V_{\max}$, i.e., if $|f(x)| > V_{\max}$ then $T(f)(x) = \text{sgn}(f(x))V_{\max}$.

To approximate a value function $V \in \mathcal{B}(\mathcal{X}; V_{\max})$, we first define a linear function space $\mathcal{F}$ spanned by the basis functions $\varphi_j \in \mathcal{B}(\mathcal{X}; L)$, $j = 1, \ldots, D$, i.e., $\mathcal{F} = \{ f_\alpha \mid f_\alpha(\cdot) = \phi(\cdot)^\top \alpha, \ \alpha \in \mathbb{R}^D \}$, where $\phi(\cdot) = \left( \varphi_1(\cdot), \ldots, \varphi_D(\cdot) \right)^\top$ is the feature vector. We define the orthogonal projection of $V$ onto the space $\mathcal{F}$ w.r.t. norm $\mu$ as $\Pi_\mathcal{F} V = \arg\min_{f \in \mathcal{F}} ||V - f||_\mu$. From $\mathcal{F}$ we can generate a $d$-dimensional ($d < D$) random space $\mathcal{G} = \{ g_\beta \mid g_\beta(\cdot) = \Psi(\cdot)^\top \beta, \ \beta \in \mathbb{R}^d \}$, where the feature vector $\Psi(\cdot) = \left( \psi_1(\cdot), \ldots, \psi_d(\cdot) \right)^\top$ is defined as $\Psi(\cdot) = A\phi(\cdot)$ with $A \in \mathbb{R}^{d \times D}$ be a random matrix whose elements are drawn i.i.d. from a suitable distribution, e.g., Gaussian $\mathcal{N}(0, 1/d)$. Similar to the space $\mathcal{F}$, we define the orthogonal projection of $V$ onto the space $\mathcal{G}$ w.r.t. norm $\mu$ as $\Pi_\mathcal{G} V = \arg\min_{g \in \mathcal{G}} ||V - g||_\mu$. Finally, for any function $f_\alpha \in \mathcal{F}$, we define $m(f_\alpha) = ||\alpha||_2 \sup_{x \in \mathcal{X}} ||\phi(x)||_2$.

## 3   LSTD with Random Projections

The objective of LSTD with random projections (LSTD-RP) is to learn the value function of a given policy from a small (relative to the dimension of the original space) number of samples in a low-dimensional linear space defined by a random projection of the high-dimensional space. We show that solving the problem in the low dimensional space instead of the original high-dimensional space reduces the estimation error at the price of a "controlled" increase in the approximation error. In this section, we introduce the notations and the resulting algorithm, and discuss its computational complexity. In Section 4, we provide the finite-sample analysis of the algorithm.

We use the linear spaces $\mathcal{F}$ and $\mathcal{G}$ with dimensions $D$ and $d$ ($d < D$) as defined in Section 2. Since in the following the policy is fixed, we drop the dependency of $R^\pi$, $P^\pi$, $V^\pi$, and $\mathcal{T}^\pi$ on $\pi$ and simply use $R$, $P$, $V$, and $\mathcal{T}$. Let $\{X_t\}_{t=1}^{n}$ be a sample path (or trajectory) of size $n$ generated by the Markov

chain $\mathcal{M}^\pi$, and let $v \in \mathbb{R}^n$ and $r \in \mathbb{R}^n$, defined as $v_t = V(X_t)$ and $r_t = R(X_t)$, be the value and reward vectors of this trajectory. Also, let $\Psi = [\Psi(X_1)^\top; \ldots; \Psi(X_n)^\top]$ be the feature matrix defined at these $n$ states and $\mathcal{G}_n = \{\Psi\beta \mid \beta \in \mathbb{R}^d\} \subset \mathbb{R}^n$ be the corresponding vector space. We denote by $\widehat{\Pi}_\mathcal{G} : \mathbb{R}^n \to \mathcal{G}_n$ the orthogonal projection onto $\mathcal{G}_n$, defined by $\widehat{\Pi}_\mathcal{G} y = \arg\min_{z \in \mathcal{G}_n} ||y - z||_n$, where $||y||_n^2 = \frac{1}{n}\sum_{t=1}^n y_t^2$. Similarly, we can define the orthogonal projection onto $\mathcal{F}_n = \{\Phi\alpha \mid \alpha \in \mathbb{R}^D\}$ as $\widehat{\Pi}_\mathcal{F} y = \arg\min_{z \in \mathcal{F}_n} ||y - z||_n$, where $\Phi = [\phi(X_1)^\top; \ldots; \phi(X_n)^\top]$ is the feature matrix defined at $\{X_t\}_{t=1}^n$. Note that for any $y \in \mathbb{R}^n$, the orthogonal projections $\widehat{\Pi}_\mathcal{G} y$ and $\widehat{\Pi}_\mathcal{F} y$ exist and are unique.

We consider the pathwise-LSTD algorithm introduced in [11]. Pathwise-LSTD takes a single trajectory $\{X_t\}_{t=1}^n$ of size $n$ generated by the Markov chain as input and returns the fixed point of the empirical operator $\widehat{\Pi}_\mathcal{G}\widehat{\mathcal{T}}$, where $\widehat{\mathcal{T}}$ is the pathwise Bellman operator defined as $\widehat{\mathcal{T}}y = r + \gamma\widehat{P}y$. The operator $\widehat{P} : \mathbb{R}^n \to \mathbb{R}^n$ is defined as $(\widehat{P}y)_t = y_{t+1}$ for $1 \le t < n$ and $(\widehat{P}y)_n = 0$. As shown in [11], $\widehat{\mathcal{T}}$ is a $\gamma$-contraction in $\ell_2$-norm, thus together with the non-expansive property of $\widehat{\Pi}_\mathcal{G}$, it guarantees the existence and uniqueness of the pathwise-LSTD fixed point $\hat{v} \in \mathbb{R}^n$, $\hat{v} = \widehat{\Pi}_\mathcal{G}\widehat{\mathcal{T}}\hat{v}$. Note that the uniqueness of $\hat{v}$ does not imply the uniqueness of the parameter $\hat{\beta}$ such that $\hat{v} = \Psi\hat{\beta}$.

---

**LSTD-RP** $\left(D, d, \{X_t\}_{t=1}^n, \{R(X_t)\}_{t=1}^n, \phi, \gamma\right)$                           **Cost**
**Compute**

- the reward vector $r_{n\times1}$ ;   $r_t = R(X_t)$                              $O(n)$
- the high-dimensional feature matrix $\Phi_{n\times D} = [\phi(X_1)^\top; \ldots; \phi(X_n)^\top]$       $O(nD)$
- the projection matrix $A_{d\times D}$ whose elements are i.i.d. samples from $\mathcal{N}(0, 1/d)$   $O(dD)$
- the low-dim feature matrix $\Psi_{n\times d} = [\Psi(X_1)^\top; \ldots; \Psi(X_n)^\top]$ ;   $\Psi(\cdot) = A\phi(\cdot)$   $O(ndD)$
- the matrix $\widehat{P}\Psi = \Psi'_{n\times d} = [\Psi(X_2)^\top; \ldots; \Psi(X_n)^\top; \mathbf{0}^\top]$               $O(nd)$
- $\tilde{A}_{d\times d} = \Psi^\top(\Psi - \gamma\Psi')$     ,        $\tilde{b}_{d\times1} = \Psi^\top r$          $O(nd + nd^2) + O(nd)$

**return** either $\hat{\beta} = \tilde{A}^{-1}\tilde{b}$   or   $\hat{\beta} = \tilde{A}^+\tilde{b}$   ($\tilde{A}^+$ is the Moore-Penrose pseudo-inverse of $\tilde{A}$)   $O(d^2 + d^3)$

---

Figure 1: The pseudo-code of the LSTD with random projections (LSTD-RP) algorithm.

Figure 1 contains the pseudo-code and the computational cost of the LSTD-RP algorithm. The total computational cost of LSTD-RP is $O(d^3 + ndD)$, while the computational cost of LSTD in the high-dimensional space $\mathcal{F}$ is $O(D^3 + nD^2)$. As we will see, the analysis of Section 4 suggests that the value of $d$ should be set to $O(\sqrt{n})$. In this case the numerical complexity of LSTD-RP is $O(n^{3/2}D)$, which is better than $O(D^3)$, the cost of LSTD in $\mathcal{F}$ when $n < D$ (the case considered in this paper). Note that the cost of making a prediction is $D$ in LSTD in $\mathcal{F}$ and $dD$ in LSTD-RP.

## 4 Finite-Sample Analysis of LSTD with Random Projections

In this section, we report the main theoretical results of the paper. In particular, we derive a performance bound for LSTD-RP in the Markov design setting, i.e., when the LSTD-RP solution is compared to the true value function only at the states belonging to the trajectory used by the algorithm (see Section 4 in [11] for a more detailed discussion). We then derive a condition on the number of samples to guarantee the uniqueness of the LSTD-RP solution. Finally, from the Markov design bound we obtain generalization bounds when the Markov chain has a stationary distribution.

### 4.1 Markov Design Bound

**Theorem 1.** *Let $\mathcal{F}$ and $\mathcal{G}$ be linear spaces with dimensions $D$ and $d$ $(d < D)$ as defined in Section 2. Let $\{X_t\}_{t=1}^n$ be a sample path generated by the Markov chain $\mathcal{M}^\pi$, and $v, \hat{v} \in \mathbb{R}^n$ be the vectors whose components are the value function and the LSTD-RP solution at $\{X_t\}_{t=1}^n$. Then for any $\delta > 0$, whenever $d \ge 15\log(8n/\delta)$, with probability $1 - \delta$ (the randomness is w.r.t. both the random sample path and the random projection), $\hat{v}$ satisfies*

$$||v - \hat{v}||_n \le \frac{1}{\sqrt{1 - \gamma^2}}\left[||v - \widehat{\Pi}_\mathcal{F} v||_n + \sqrt{\frac{8\log(8n/\delta)}{d}}\, m(\widehat{\Pi}_\mathcal{F} v)\right] + \frac{\gamma V_{\max} L}{1 - \gamma}\sqrt{\frac{d}{\nu_n}}\left(\sqrt{\frac{8\log(4d/\delta)}{n}} + \frac{1}{n}\right),$$

(1)

*where the random variable $\nu_n$ is the smallest strictly positive eigenvalue of the sample-based Gram matrix $\frac{1}{n}\Psi^\top\Psi$. Note that $m(\widehat{\Pi}_{\mathcal{F}}v) = m(f_\alpha)$ with $f_\alpha$ be any function in $\mathcal{F}$ such that $f_\alpha(X_t) = (\widehat{\Pi}_{\mathcal{F}}v)_t$ for $1 \le t \le n$.*

Before stating the proof of Theorem 1, we need to prove the following lemma.

**Lemma 1.** *Let $\mathcal{F}$ and $\mathcal{G}$ be linear spaces with dimensions $D$ and $d$ ($d < D$) as defined in Section 2. Let $\{X_i\}_{i=1}^n$ be $n$ states and $f_\alpha \in \mathcal{F}$. Then for any $\delta > 0$, whenever $d \ge 15\log(4n/\delta)$, with probability $1 - \delta$ (the randomness is w.r.t. the random projection), we have*

$$\inf_{g \in \mathcal{G}} ||f_\alpha - g||_n^2 \le \frac{8\log(4n/\delta)}{d} m(f_\alpha)^2. \tag{2}$$

*Proof.* The proof relies on the application of a variant of Johnson-Lindenstrauss (JL) lemma which states that the inner-products are approximately preserved by the application of the random matrix $A$ (see e.g., Proposition 1 in [14]). For any $\delta > 0$, we set $\epsilon^2 = \frac{8}{d}\log(4n/\delta)$. Thus for $d \ge 15\log(4n/\delta)$, we have $\epsilon \le 3/4$ and as a result $\epsilon^2/4 - \epsilon^3/6 \ge \epsilon^2/8$ and $d \ge \frac{\log(4n/\delta)}{\epsilon^2/4 - \epsilon^3/6}$. Thus, from Proposition 1 in [14], for all $1 \le i \le n$, we have $|\phi(X_i) \cdot \alpha - A\phi(X_i) \cdot A\alpha| \le \epsilon ||\alpha||_2 ||\phi(X_i)||_2 \le \epsilon\, m(f_\alpha)$ with high probability. From this result, we deduce that with probability $1 - \delta$

$$\inf_{g \in \mathcal{G}} ||f_\alpha - g||_n^2 \le ||f_\alpha - g_{A\alpha}||_n^2 = \frac{1}{n}\sum_{i=1}^n |\phi(X_i) \cdot \alpha - A\phi(X_i) \cdot A\alpha|^2 \le \frac{8\log(4n/\delta)}{d} m(f_\alpha)^2.$$

$\square$

*Proof of Theorem 1.* For any fixed space $\mathcal{G}$, the performance of the LSTD-RP solution can be bounded according to Theorem 1 in [10] as

$$||v - \hat{v}||_n \le \frac{1}{\sqrt{1-\gamma^2}}||v - \widehat{\Pi}_{\mathcal{G}}v||_n + \frac{\gamma V_{\max}L}{1-\gamma}\sqrt{\frac{d}{\nu_n}}\Big(\sqrt{\frac{8\log(2d/\delta')}{n}} + \frac{1}{n}\Big), \tag{3}$$

with probability $1 - \delta'$ (w.r.t. the random sample path). From the triangle inequality, we have

$$||v - \widehat{\Pi}_{\mathcal{G}}v||_n \le ||v - \widehat{\Pi}_{\mathcal{F}}v||_n + ||\widehat{\Pi}_{\mathcal{F}}v - \widehat{\Pi}_{\mathcal{G}}v||_n = ||v - \widehat{\Pi}_{\mathcal{F}}v||_n + ||\widehat{\Pi}_{\mathcal{F}}v - \widehat{\Pi}_{\mathcal{G}}(\widehat{\Pi}_{\mathcal{F}}v)||_n. \tag{4}$$

The equality in Eq. 4 comes from the fact that for any vector $g \in \mathcal{G}$, we can write $||v - g||_n^2 = ||v - \widehat{\Pi}_{\mathcal{F}}v||_n^2 + ||\widehat{\Pi}_{\mathcal{F}}v - g||_n^2$. Since $||v - \widehat{\Pi}_{\mathcal{F}}v||_n$ is independent of $g$, we have $\arg\inf_{g \in \mathcal{G}}||v - g||_n^2 = \arg\inf_{g \in \mathcal{G}}||\widehat{\Pi}_{\mathcal{F}}v - g||_n^2$, and thus, $\widehat{\Pi}_{\mathcal{G}}v = \widehat{\Pi}_{\mathcal{G}}(\widehat{\Pi}_{\mathcal{F}}v)$. From Lemma 1, if $d \ge 15\log(4n/\delta'')$, with probability $1 - \delta''$ (w.r.t. the choice of $A$), we have

$$||\widehat{\Pi}_{\mathcal{F}}v - \widehat{\Pi}_{\mathcal{G}}(\widehat{\Pi}_{\mathcal{F}}v)||_n \le \sqrt{\frac{8\log(4n/\delta'')}{d}} m(\widehat{\Pi}_{\mathcal{F}}v). \tag{5}$$

We conclude from a union bound argument that Eqs. 3 and 5 hold simultaneously with probability at least $1 - \delta' - \delta''$. The claim follows by combining Eqs. 3–5, and setting $\delta' = \delta'' = \delta/2$. $\square$

**Remark 1.** Using Theorem 1, we can compare the performance of LSTD-RP with the performance of LSTD directly applied in the high-dimensional space $\mathcal{F}$. Let $\bar{v}$ be the LSTD solution in $\mathcal{F}$, then up to constants, logarithmic, and dominated factors, with high probability, $\bar{v}$ satisfies

$$||v - \bar{v}||_n \le \frac{1}{\sqrt{1-\gamma^2}}||v - \widehat{\Pi}_{\mathcal{F}}v||_n + \frac{1}{1-\gamma}O(\sqrt{D/n}). \tag{6}$$

By comparing Eqs. 1 and 6, we notice that **1)** the estimation error of $\hat{v}$ is of order $O(\sqrt{d/n})$, and thus, is smaller than the estimation error of $\bar{v}$, which is of order $O(\sqrt{D/n})$, and **2)** the approximation error of $\hat{v}$ is the approximation error of $\bar{v}$, $||v - \widehat{\Pi}_{\mathcal{F}}v||_n$, plus an additional term that depends on $m(\widehat{\Pi}_{\mathcal{F}}v)$ and decreases with $d$, the dimensionality of $\mathcal{G}$, with the rate $O(\sqrt{1/d})$. Hence, LSTD-RP may have a better performance than solving LSTD in $\mathcal{F}$ whenever this additional term is smaller than the gain achieved in the estimation error. Note that $m(\widehat{\Pi}_{\mathcal{F}}v)$ highly depends on the value function $V$ that is being approximated and the features of the space $\mathcal{F}$. It is important to carefully tune the value of $d$ as both the estimation error and the additional approximation error in Eq. 1 depend on $d$. For instance, while a small value of $d$ significantly reduces the estimation error (and the need for samples), it may amplify the additional approximation error term, and thus, reduce the advantage of LSTD-RP over LSTD. We may get an idea on how to select the value of $d$ by optimizing the bound

$$d = \frac{m(\widehat{\Pi}_{\mathcal{F}}v)}{\gamma V_{\max}L}\sqrt{\frac{n\nu_n(1-\gamma)}{1+\gamma}}. \tag{7}$$

Therefore, when $n$ samples are available the optimal value for $d$ is of the order $O(\sqrt{n})$. Using the value of $d$ in Eq. 7, we can rewrite the bound of Eq. 1 as (up to the dominated term $1/n$)

$$||v - \hat{v}||_n \leq \frac{1}{\sqrt{1-\gamma^2}}||v - \widehat{\Pi}_{\mathcal{F}}v||_n + \frac{1}{1-\gamma}\sqrt{8\log(8n/\delta)}\sqrt{\gamma V_{\max}L\, m(\widehat{\Pi}_{\mathcal{F}}v)}\Big(\frac{1-\gamma}{n\nu_n(1+\gamma)}\Big)^{1/4}. \tag{8}$$

Using Eqs. 6 and 8, it would be easier to compare the performance of LSTD-RP and LSTD in space $\mathcal{F}$, and observe the role of the term $m(\widehat{\Pi}_{\mathcal{F}}v)$. For further discussion on $m(\widehat{\Pi}_{\mathcal{F}}v)$ refer to [14] and for the case of $D = \infty$ to Section 4.3 of this paper.

**Remark 2.** As discussed in the introduction, when the dimensionality $D$ of $\mathcal{F}$ is much bigger than the number of samples $n$, the learning algorithms are likely to overfit the data. In this case, it is reasonable to assume that the target vector $v$ itself belongs to the vector space $\mathcal{F}_n$. We state this condition using the following assumption:

**Assumption 1.** *(Overfitting). For any set of $n$ points $\{X_i\}_{i=1}^n$, there exists a function $f \in \mathcal{F}$ such that $f(X_i) = V(X_i)$, $1 \leq i \leq n$.*

Assumption 1 is equivalent to require that the rank of the empirical Gram matrices $\frac{1}{n}\Phi^\top\Phi$ to be bigger than $n$. Note that Assumption 1 is likely to hold whenever $D \gg n$, because in this case we can expect that the features to be independent enough on $\{X_i\}_{i=1}^n$ so that the rank of $\frac{1}{n}\Phi^\top\Phi$ to be bigger than $n$ (e.g., if the features are linearly independent on the samples, it is sufficient to have $D \geq n$). Under Assumption 1 we can remove the empirical approximation error term in Theorem 1 and deduce the following result.

**Corollary 1.** *Under Assumption 1 and the conditions of Theorem 1, with probability $1-\delta$ (w.r.t. the random sample path and the random space), $\hat{v}$ satisfies*

$$||v - \hat{v}||_n \leq \frac{1}{\sqrt{1-\gamma^2}}\sqrt{\frac{8\log(8n/\delta)}{d}}m(\widehat{\Pi}_{\mathcal{F}}v) + \frac{\gamma V_{\max}L}{1-\gamma}\sqrt{\frac{d}{\nu_n}}\Big(\sqrt{\frac{8\log(4d/\delta)}{n}} + \frac{1}{n}\Big).$$

### 4.2 Uniqueness of the LSTD-RP Solution

While the results in the previous section hold for any Markov chain, in this section we assume that the Markov chain $\mathcal{M}^\pi$ admits a stationary distribution $\rho$ and is exponentially fast $\beta$-mixing with parameters $\bar{\beta}, b, \kappa$, i.e., its $\beta$-mixing coefficients satisfy $\beta_i \leq \bar{\beta}\exp(-bi^\kappa)$ (see e.g., Sections 8.2 and 8.3 in [10] for a more detailed definition of $\beta$-mixing processes). As shown in [11, 10], if $\rho$ exists, it would be possible to derive a condition for the existence and uniqueness of the LSTD solution depending on the number of samples and the smallest eigenvalue of the Gram matrix defined according to the stationary distribution $\rho$, i.e., $G \in \mathbb{R}^{D \times D}$, $G_{ij} = \int \varphi_i(x)\varphi_j(x)\rho(dx)$. We now discuss the existence and uniqueness of the LSTD-RP solution. Note that as $D$ increases, the smallest eigenvalue of $G$ is likely to become smaller and smaller. In fact, the more the features in $\mathcal{F}$, the higher the chance for some of them to be correlated under $\rho$, thus leading to an ill-conditioned matrix $G$. On the other hand, since $d < D$, the probability that $d$ independent random combinations of $\varphi_i$ lead to highly correlated features $\psi_j$ is relatively small. In the following we prove that the smallest eigenvalue of the Gram matrix $H \in \mathbb{R}^{d \times d}$, $H_{ij} = \int \psi_i(x)\psi_j(x)\rho(dx)$ in the random space $\mathcal{G}$ is indeed bigger than the smallest eigenvalue of $G$ with high probability.

**Lemma 2.** *Let $\delta > 0$ and $\mathcal{F}$ and $\mathcal{G}$ be linear spaces with dimensions $D$ and $d$ $(d < D)$ as defined in Section 2 with $D > d + 2\sqrt{2d\log(2/\delta)} + 2\log(2/\delta)$. Let the elements of the projection matrix $A$ be Gaussian random variables drawn from $\mathcal{N}(0, 1/d)$. Let the Markov chain $\mathcal{M}^\pi$ admit a stationary distribution $\rho$. Let $G$ and $H$ be the Gram matrices according to $\rho$ for the spaces $\mathcal{F}$ and $\mathcal{G}$, and $\omega$ and $\chi$ be their smallest eigenvalues. We have with probability $1-\delta$ (w.r.t. the random space)*

$$\chi \geq \frac{D}{d}\omega\left(1 - \sqrt{\frac{d}{D}} - \sqrt{\frac{2\log(2/\delta)}{D}}\right)^2. \tag{9}$$

*Proof.* Let $\beta \in \mathbb{R}^d$ be the eigenvector associated to the smallest eigenvalue $\chi$ of $H$, from the definition of the features $\Psi$ of $\mathcal{G}$ $(H = AGA^\top)$ and linear algebra, we obtain

$$\chi||\beta||_2^2 = \beta^\top \chi \beta = \beta^\top H \beta = \beta^\top A G A^\top \beta \geq \omega ||A^\top \beta||_2^2 = \omega\, \beta^\top A A^\top \beta \geq \omega\, \xi\, ||\beta||_2^2\,, \tag{10}$$

where $\xi$ is the smallest eigenvalue of the random matrix $AA^\top$, or in other words, $\sqrt{\xi}$ is the smallest singular value of the $D \times d$ random matrix $A^\top$, i.e., $s_{\min}(A^\top) = \sqrt{\xi}$. We now define $B = \sqrt{d}A$. Note that if the elements of $A$ are drawn from the Gaussian distribution $\mathcal{N}(0, 1/d)$, the elements of $B$ are standard Gaussian random variables, and thus, the smallest eigenvalue of $AA^\top$, $\xi$, can be written as $\xi = s_{\min}^2(B^\top)/d$. There has been extensive work on extreme singular values of random matrices (see e.g., [19]). For a $D \times d$ random matrix with independent standard normal random variables, such as $B^\top$, we have with probability $1 - \delta$ (see [19] for more details)

$$s_{\min}(B^\top) \geq \left( \sqrt{D} - \sqrt{d} - \sqrt{2\log(2/\delta)} \right). \tag{11}$$

From Eq. 11 and the relation between $\xi$ and $s_{\min}(B^\top)$, we obtain

$$\xi \geq \frac{D}{d} \left( 1 - \sqrt{\frac{d}{D}} - \sqrt{\frac{2\log(2/\delta)}{D}} \right)^2, \tag{12}$$

with probability $1 - \delta$. The claim follows by replacing the bound for $\xi$ from Eq. 12 in Eq. 10. $\qquad\square$

The result of Lemma 2 is for Gaussian random matrices. However, it would be possible to extend this result using non-asymptotic bounds for the extreme singular values of more general random matrices [19]. Note that in Eq. 9, $D/d$ is always greater than 1 and the term in the parenthesis approaches 1 for large values of $D$. Thus, we can conclude that with high probability the smallest eigenvalue $\chi$ of the Gram matrix $H$ of the randomly generated low-dimensional space $\mathcal{G}$ is bigger than the smallest eigenvalue $\omega$ of the Gram matrix $G$ of the high-dimensional space $\mathcal{F}$.

**Lemma 3.** *Let $\delta > 0$ and $\mathcal{F}$ and $\mathcal{G}$ be linear spaces with dimensions $D$ and $d$ ($d < D$) as defined in Section 2 with $D > d + 2\sqrt{2d\log(2/\delta)} + 2\log(2/\delta)$. Let the elements of the projection matrix $A$ be Gaussian random variables drawn from $\mathcal{N}(0, 1/d)$. Let the Markov chain $\mathcal{M}^\pi$ admit a stationary distribution $\rho$. Let $G$ be the Gram matrix according to $\rho$ for space $\mathcal{F}$ and $\omega$ be its smallest eigenvalue. Let $\{X_t\}_{t=1}^n$ be a trajectory of length $n$ generated by a stationary $\beta$-mixing process with stationary distribution $\rho$. If the number of samples $n$ satisfies*

$$n > \frac{288 L^2\, d\, \Lambda(n, d, \delta/2)}{\omega D}\, \max\left\{ \frac{\Lambda(n, d, \delta/2)}{b}, 1 \right\}^{1/\kappa} \left( 1 - \sqrt{\frac{d}{D}} - \sqrt{\frac{2\log(2/\delta)}{D}} \right)^{-2}, \tag{13}$$

*where $\Lambda(n, d, \delta) = 2(d+1)\log n + \log\frac{e}{\delta} + \log^+\left( \max\{18(6e)^{2(d+1)}, \bar\beta\} \right)$, then with probability $1 - \delta$, the features $\psi_1, \ldots, \psi_d$ are linearly independent on the states $\{X_t\}_{t=1}^n$, i.e., $||g_\beta||_n = 0$ implies $\beta = 0$, and the smallest eigenvalue $\nu_n$ of the sample-based Gram matrix $\frac{1}{n}\Psi^\top \Psi$ satisfies*

$$\sqrt{\nu_n} \geq \sqrt{\nu} = \frac{\sqrt{\omega}}{2} \sqrt{\frac{D}{d}} \left( 1 - \sqrt{\frac{d}{D}} - \sqrt{\frac{2\log(\frac{2}{\delta})}{D}} \right) - 6L \sqrt{\frac{2\Lambda(n, d, \frac{\delta}{2})}{n}\, \max\left\{ \frac{\Lambda(n, d, \frac{\delta}{2})}{b}, 1 \right\}^{1/\kappa}} > 0\,. \tag{14}$$

*Proof.* The proof follows similar steps as in Lemma 4 in [10]. A sketch of the proof is available in [6]. $\qquad\square$

By comparing Eq. 13 with Eq. 13 in [10], we can see that the number of samples needed for the empirical Gram matrix $\frac{1}{n}\Psi^\top \Psi$ in $\mathcal{G}$ to be invertible with high probability is less than that for its counterpart $\frac{1}{n}\Phi^\top \Phi$ in the high-dimensional space $\mathcal{F}$.

### 4.3 Generalization Bound

In this section, we show how Theorem 1 can be generalized to the entire state space $\mathcal{X}$ when the Markov chain $\mathcal{M}^\pi$ has a stationary distribution $\rho$. We consider the case in which the samples $\{X_t\}_{t=1}^n$ are obtained by following a single trajectory in the stationary regime of $\mathcal{M}^\pi$, i.e., when $X_1$ is drawn from $\rho$. As discussed in Remark 2 of Section 4.1, it is reasonable to assume that the high-dimensional space $\mathcal{F}$ contains functions that are able to perfectly fit the value function $V$ in any finite number $n$ ($n < D$) of states $\{X_t\}_{t=1}^n$, thus we state the following theorem under Assumption 1.

**Theorem 2.** *Let $\delta > 0$ and $\mathcal{F}$ and $\mathcal{G}$ be linear spaces with dimensions $D$ and $d$ $(d < D)$ as defined in Section 2 with $d \geq 15 \log(8n/\delta)$. Let $\{X_t\}_{t=1}^n$ be a path generated by a stationary $\beta$-mixing process with stationary distribution $\rho$. Let $\hat{V}$ be the LSTD-RP solution in the random space $\mathcal{G}$. Then under Assumption 1, with probability $1 - \delta$ (w.r.t. the random sample path and the random space),*

$$||V - T(\hat{V})||_\rho \leq \frac{2}{\sqrt{1-\gamma^2}} \sqrt{\frac{8\log(24n/\delta)}{d}} m(\Pi_\mathcal{F} V) + \frac{2\gamma V_{\max} L}{1-\gamma} \sqrt{\frac{d}{\nu}} \left( \sqrt{\frac{8\log(12d/\delta)}{n}} + \frac{1}{n} \right) + \epsilon \ , \quad (15)$$

*where $\nu$ is a lower bound on the eigenvalues of the Gram matrix $\frac{1}{n}\Psi^\top\Psi$ defined by Eq. 14 and*

$$\epsilon = 24 V_{\max} \sqrt{\frac{2\Lambda(n, d, \delta/3)}{n} \max\left\{ \frac{\Lambda(n, d, \delta/3)}{b}, 1 \right\}^{1/\kappa}} \ .$$

*with $\Lambda(n, d, \delta)$ defined as in Lemma 3. Note that $T$ in Eq. 15 is the truncation operator defined in Section 2.*

*Proof.* The proof is a consequence of applying concentration of measures inequalities for $\beta$-mixing processes and linear spaces (see Corollary 18 in [10]) on the term $||V - T(\hat{V})||_n$, using the fact that $||V - T(\hat{V})||_n \leq ||V - \hat{V}||_n$, and using the bound of Corollary 1. The bound of Corollary 1 and the lower bound on $\nu$, each one holding with probability $1 - \delta'$, thus, the statement of the theorem (Eq. 15) holds with probability $1 - \delta$ by setting $\delta = 3\delta'$. $\qquad\qquad\square$

**Remark 1.** An interesting property of the bound in Theorem 2 is that the approximation error of $V$ in space $\mathcal{F}$, $||V - \Pi_\mathcal{F} V||_\rho$, does not appear and the error of the LSTD solution in the randomly projected space only depends on the dimensionality $d$ of $\mathcal{G}$ and the number of samples $n$. However this property is valid only when Assumption 1 holds, i.e., at most for $n \leq D$. An interesting case here is when the dimension of $\mathcal{F}$ is infinite ($D = \infty$), so that the bound is valid for any number of samples $n$. In [15], two approximation spaces $\mathcal{F}$ of infinite dimension were constructed based on a multi-resolution set of features that are rescaled and translated versions of a given mother function. In the case that the mother function is a wavelet, the resulting features, called scrambled wavelets, are linear combinations of wavelets at all scales weighted by Gaussian coefficients. As a results, the corresponding approximation space is a Sobolev space $H^s(\mathcal{X})$ with smoothness of order $s > p/2$, where $p$ is the dimension of the state space $\mathcal{X}$. In this case, for a function $f_\alpha \in H^s(\mathcal{X})$, it is proved that the $\ell_2$-norm of the parameter $\alpha$ is equal to the norm of the function in $H^s(\mathcal{X})$, i.e., $||\alpha||_2 = ||f_\alpha||_{H^s(\mathcal{X})}$. We do not describe those results further and refer the interested readers to [15]. What is important about the results of [15] is that it shows that it is possible to consider infinite dimensional function spaces for which $\sup_x ||\phi(x)||_2$ is finite and $||\alpha||_2$ is expressed in terms of the norm of $f_\alpha$ in $\mathcal{F}$. In such cases, $m(\Pi_\mathcal{F} V)$ is finite and the bound of Theorem 2, which does not contain any approximation error of $V$ in $\mathcal{F}$, holds for any $n$. Nonetheless, further investigation is needed to better understand the role of $||f_\alpha||_{H^s(\mathcal{X})}$ in the final bound.

**Remark 2.** As discussed in the introduction, regularization methods have been studied in solving high-dimensional RL problems. Therefore, it is interesting to compare our results for LSTD-RP with those reported in [4] for $\ell_2$-regularized LSTD. Under Assumption 1, when $D = \infty$, by selecting the features as described in the previous remark and optimizing the value of $d$ as in Eq. 7, we obtain

$$||V - T(\hat{V})||_\rho \leq O\left( \sqrt{||f_\alpha||_{H^s(\mathcal{X})}}\, n^{-1/4} \right). \quad (16)$$

Although the setting considered in [4] is different than ours (e.g., the samples are i.i.d.), a qualitative comparison of Eq. 16 with the bound in Theorem 2 of [4] shows a striking similarity in the performance of the two algorithms. In fact, they both contain the Sobolev norm of the target function and have a similar dependency on the number of samples with a convergence rate of $O(n^{-1/4})$ (when the smoothness of the Sobolev space in [4] is chosen to be half of the dimensionality of $\mathcal{X}$). This similarity asks for further investigation on the difference between $\ell_2$-regularized methods and random projections in terms of prediction performance and computational complexity.

## 5 LSPI with Random Projections

In this section, we move from policy evaluation to policy iteration and provide a performance bound for LSPI with random projections (LSPI-RP), i.e., a policy iteration algorithm that uses LSTD-RP at each iteration. LSPI-RP starts with an arbitrary initial value function $V_{-1} \in \mathcal{B}(\mathcal{X}; V_{\max})$ and its corresponding greedy policy $\pi_0$. At the first iteration, it approximates $V^{\pi_0}$ using LSTD-RP and

returns a function $\hat{V}_0$, whose truncated version $\tilde{V}_0 = T(\hat{V}_0)$ is used to build the policy for the second iteration. More precisely, $\pi_1$ is a greedy policy w.r.t. $\tilde{V}_0$. So, at each iteration $k$, a function $\hat{V}_{k-1}$ is computed as an approximation to $V^{\pi_{k-1}}$, and then truncated, $\tilde{V}_{k-1}$, and used to build the policy $\pi_k$.[1] Note that in general, the measure $\sigma \in \mathcal{S}(\mathcal{X})$ used to evaluate the final performance of the LSPI-RP algorithm might be different from the distribution used to generate samples at each iteration. Moreover, the LSTD-RP performance bounds require the samples to be collected by following the policy under evaluation. Thus, we need Assumptions 1-3 in [10] in order to **1)** define a lower-bounding distribution $\mu$ with constant $C < \infty$, **2)** guarantee that with high probability a unique LSTD-RP solution exists at each iteration, and **3)** define the slowest $\beta$-mixing process among all the mixing processes $\mathcal{M}^{\pi_k}$ with $0 \le k < K$.

**Theorem 3.** *Let $\delta > 0$ and $\mathcal{F}$ and $\mathcal{G}$ be linear spaces with dimensions $D$ and $d$ ($d < D$) as defined in Section 2 with $d \ge 15 \log(8Kn/\delta)$. At each iteration $k$, we generate a path of size $n$ from the stationary $\beta$-mixing process with stationary distribution $\rho_{k-1} = \rho^{\pi_{k-1}}$. Let $n$ satisfy the condition in Eq. 13 for the slower $\beta$-mixing process. Let $V_{-1}$ be an arbitrary initial value function, $\hat{V}_0, \dots, \hat{V}_{K-1}$ ($\tilde{V}_0, \dots, \tilde{V}_{K-1}$) be the sequence of value functions (truncated value functions) generated by LSPI-RP, and $\pi_K$ be the greedy policy w.r.t. $\tilde{V}_{K-1}$. Then, under Assumption 1 and Assumptions 1-3 in [10], with probability $1 - \delta$ (w.r.t. the random samples and the random spaces), we have*

$$||V^* - V^{\pi_K}||_\sigma \le \frac{4\gamma}{(1-\gamma)^2}\left\{(1+\gamma)\sqrt{CC_{\sigma,\mu}}\left[\frac{2V_{\max}}{\sqrt{1-\gamma^2}}\sqrt{\frac{C}{\omega_\mu}}\sqrt{\frac{8\log(24Kn/\delta)}{d}}\sup_{x\in\mathcal{X}}||\phi(x)||_2\right.\right. \quad (17)$$

$$\left.\left. + \frac{2\gamma V_{\max}L}{1-\gamma}\sqrt{\frac{d}{\nu_\mu}}\Big(\sqrt{\frac{8\log(12Kd/\delta)}{n}} + \frac{1}{n}\Big) + \mathcal{E}\right] + \gamma^{\frac{K-1}{2}}R_{\max}\right\},$$

*where $C_{\sigma,\mu}$ is the concentrability term from Definition 2 in [1], $\omega_\mu$ is the smallest eigenvalue of the Gram matrix of space $\mathcal{F}$ w.r.t. $\mu$, $\nu_\mu$ is $\nu$ from Eq. 14 in which $\omega$ is replaced by $\omega_\mu$, and $\mathcal{E}$ is $\epsilon$ from Theorem 2 written for the slowest $\beta$-mixing process.*

*Proof.* The proof follows similar lines as in the proof of Thm. 8 in [10] and is available in [6]. $\square$

**Remark**. The most critical issue about Theorem 3 is the validity of Assumptions 1-3 in [10]. It is important to note that Assumption 1 is needed to bound the performance of LSPI independent from the use of random projections (see [10]). On the other hand, Assumption 2 is explicitly related to random projections and allows us to bound the term $m(\Pi_\mathcal{F}V)$. In order for this assumption to hold, the features $\{\varphi_j\}_{j=1}^D$ of the high-dimensional space $\mathcal{F}$ should be carefully chosen so as to be linearly independent w.r.t. $\mu$.

## 6 Conclusions

Learning in high-dimensional linear spaces is particularly appealing in RL because it allows to have a very accurate approximation of value functions. Nonetheless, the larger the space, the higher the need of samples and the risk of overfitting. In this paper, we introduced an algorithm, called LSTD-RP, in which LSTD is run in a low-dimensional space obtained by a random projection of the original high-dimensional space. We theoretically analyzed the performance of LSTD-RP and showed that it solves the problem of overfitting (i.e., the estimation error depends on the value of the low dimension) at the cost of a slight worsening in the approximation accuracy compared to the high-dimensional space. We also analyzed the performance of LSPI-RP, a policy iteration algorithm that uses LSTD-RP for policy evaluation. The analysis reported in the paper opens a number of interesting research directions such as: **1)** comparison of LSTD-RP to $\ell_2$ and $\ell_1$ regularized approaches, and **2)** a thorough analysis of the case when $D = \infty$ and the role of $||f_\alpha||_{H^s(\mathcal{X})}$ in the bound.

**Acknowledgments** This work was supported by French National Research Agency through the projects EXPLO-RA $n^\circ$ ANR-08-COSI-004 and LAMPADA $n^\circ$ ANR-09-EMER-007, by Ministry of Higher Education and Research, Nord-Pas de Calais Regional Council and FEDER through the "contrat de projets état region 2007–2013", and by PASCAL2 European Network of Excellence.

## Footnotes

[1]Note that the MDP model is needed to generate a greedy policy $\pi_k$. In order to avoid the need for the model, we can simply move to LSTD-$Q$ with random projections. Although the analysis of LSTD-RP can be extended to action-value functions and LSTD-RP-$Q$, for simplicity we use value functions in the following.

# References

[1] A. Antos, Cs. Szepesvari, and R. Munos. Learning near-optimal policies with Bellman-residual minimization based fitted policy iteration and a single sample path. *Machine Learning Journal*, 71:89–129, 2008.

[2] J. Boyan. Least-squares temporal difference learning. *Proceedings of the 16th International Conference on Machine Learning*, pages 49–56, 1999.

[3] S. Bradtke and A. Barto. Linear least-squares algorithms for temporal difference learning. *Machine Learning*, 22:33–57, 1996.

[4] A. M. Farahmand, M. Ghavamzadeh, Cs. Szepesvári, and S. Mannor. Regularized policy iteration. In *Proceedings of Advances in Neural Information Processing Systems 21*, pages 441–448. MIT Press, 2008.

[5] A. M. Farahmand, M. Ghavamzadeh, Cs. Szepesvári, and S. Mannor. Regularized fitted Q-iteration for planning in continuous-space Markovian decision problems. In *Proceedings of the American Control Conference*, pages 725–730, 2009.

[6] M. Ghavamzadeh, A. Lazaric, O. Maillard, and R. Munos. LSPI with random projections. Technical Report inria-00530762, INRIA, 2010.

[7] P. Keller, S. Mannor, and D. Precup. Automatic basis function construction for approximate dynamic programming and reinforcement learning. In *Proceedings of the Twenty-Third International Conference on Machine Learning*, pages 449–456, 2006.

[8] Z. Kolter and A. Ng. Regularization and feature selection in least-squares temporal difference learning. In *Proceedings of the Twenty-Sixth International Conference on Machine Learning*, pages 521–528, 2009.

[9] M. Lagoudakis and R. Parr. Least-squares policy iteration. *Journal of Machine Learning Research*, 4:1107–1149, 2003.

[10] A. Lazaric, M. Ghavamzadeh, and R. Munos. Finite-sample analysis of least-squares policy iteration. Technical Report inria-00528596, INRIA, 2010.

[11] A. Lazaric, M. Ghavamzadeh, and R. Munos. Finite-sample analysis of LSTD. In *Proceedings of the Twenty-Seventh International Conference on Machine Learning*, pages 615–622, 2010.

[12] M. Loth, M. Davy, and P. Preux. Sparse temporal difference learning using lasso. In *IEEE Symposium on Approximate Dynamic Programming and Reinforcement Learning*, pages 352–359, 2007.

[13] S. Mahadevan. Representation policy iteration. In *Proceedings of the Twenty-First Conference on Uncertainty in Artificial Intelligence*, pages 372–379, 2005.

[14] O. Maillard and R. Munos. Compressed least-squares regression. In *Proceedings of Advances in Neural Information Processing Systems 22*, pages 1213–1221, 2009.

[15] O. Maillard and R. Munos. Brownian motions and scrambled wavelets for least-squares regression. Technical Report inria-00483014, INRIA, 2010.

[16] I. Menache, S. Mannor, and N. Shimkin. Basis function adaptation in temporal difference reinforcement learning. *Annals of Operations Research*, 134:215–238, 2005.

[17] R. Parr, C. Painter-Wakefield, L. Li, and M. Littman. Analyzing feature generation for value-function approximation. In *Proceedings of the Twenty-Fourth International Conference on Machine Learning*, pages 737–744, 2007.

[18] M. Petrik, G. Taylor, R. Parr, and S. Zilberstein. Feature selection using regularization in approximate linear programs for Markov decision processes. In *Proceedings of the Twenty-Seventh International Conference on Machine Learning*, pages 871–878, 2010.

[19] M. Rudelson and R. Vershynin. Non-asymptotic theory of random matrices: extreme singular values. In *Proceedings of the International Congress of Mathematicians*, 2010.

[20] R. Sutton and A. Barto. *Reinforcement Learning: An Introduction*. MIP Press, 1998.

[21] S. Vempala. *The Random Projection Method*. American Mathematical Society, 2004.

